# Epitome driven 3-D Diffusion Tensor image segmentation: on extracting *specific* structures*

**Kamiya Motwani**[†§]   **Nagesh Adluru**[§]   **Chris Hinrichs**[†§]   **Andrew Alexander**[‡]   **Vikas Singh**[§†]

[†]**Computer Sciences**        [§]**Biostatistics & Medical Informatics**        [‡]**Medical Physics**
**University of Wisconsin**        **University of Wisconsin**        **University of Wisconsin**

{kmotwani,hinrichs,vsingh}@cs.wisc.edu   {adluru,alalexander2}@wisc.edu

## Abstract

We study the problem of segmenting *specific* white matter structures of interest from Diffusion Tensor (DT-MR) images of the human brain. This is an important requirement in many Neuroimaging studies: for instance, to evaluate whether a brain structure exhibits group level differences as a function of disease in a set of images. Typically, interactive expert guided segmentation has been the method of choice for such applications, but this is tedious for large datasets common today. To address this problem, we endow an image segmentation algorithm with "advice" encoding some global characteristics of the region(s) we want to extract. This is accomplished by constructing (using expert-segmented images) an *epitome* of a specific region – as a histogram over a bag of 'words' (e.g., suitable feature descriptors). Now, given such a representation, the problem reduces to segmenting a new brain image with additional constraints that enforce consistency between the segmented foreground and the pre-specified histogram over features. We present combinatorial approximation algorithms to incorporate such domain specific constraints for Markov Random Field (MRF) segmentation. Making use of recent results on image co-segmentation, we derive effective solution strategies for our problem. We provide an analysis of solution quality, and present promising experimental evidence showing that many structures of interest in Neuroscience can be extracted reliably from 3-D brain image volumes using our algorithm.

## 1   Introduction

Diffusion Tensor Imaging (DTI or DT-MR) is an imaging modality that facilitates measurement of the diffusion of water molecules in tissues. DTI has turned out to be especially useful in Neuroimaging because the inherent microstructure and connectivity networks in the brain can be estimated from such data [1]. The primary motivation is to investigate how specific components (i.e., structures) of the brain network topology respond to disease and treatment [2], and how these are affected as a result of external factors such as trauma. An important challenge here is to reliably extract (i.e., segment) specific structures of interest from DT-MR image volumes, so that these regions can then be analyzed to evaluate variations between clinically disparate groups. This paper focuses on efficient algorithms for this application – that is, 3-D image segmentation with side constraints to preserve fidelity of the extracted foreground with a given *epitome* of the brain region of interest.

DTI data are represented as a $3 \times 3$ positive semidefinite tensor at each image voxel. These images provide information about connection pathways in the brain, and neuroscientists focus on the

analysis of white-matter regions (these are known to encompass the 'brain axonal networks'). In general, standard segmentation methods yield reasonable results in separating white matter (WM) from gray-matter (GM), see [3]. While some of these algorithms make use of the tensor field directly [4], others utilize 'maps' of certain scalar-valued anisotropy measures calculated from tensors to partition WM/GM regions [5], see Fig. 1. But *different* pathways play *different* functional roles; hence it is more meaningful to evaluate group differences in a population at the level of *specific* white matter structures (e.g., corpus callosum, fornix, cingulum bundle). Part of the reason is that even significant volume differences in small structures may be overwhelmed in a pair-wise $t$-test using volume measures of the *entire* white matter (obtained via WM/GM segmentation [6]). To analyze variations in specific regions, we require segmentation of such structures as a first step.

Unsupervised segmentation of specific regions of interest from DTI is difficult. Even interactive segmentation (based on gray-level fractional anisotropy maps) leads to unsatisfactory results unless guided by a neuroanatomical expert – that is, specialized knowledge of the global appearance of the structure is essential in this process. Further, this is tedious for large datasets. One alternative is to use a set of already segmented images to facilitate processing of new data. Fortunately, since many studies use hand indicated regions for group analysis [7], such data *is* readily available. However, directly applying off the shelf toolboxes to learn a classifier (from such segmented images) does not work well. Part of the reason is that the local spatial context at each tensor voxel, while useful, is not sufficiently discriminative. In fact, the likelihood of a voxel to be assigned as part of the foreground (structure of interest) depends on whether the set of all foreground voxels (in entirety) *match* an 'appearance model' of the structure, in addition to being perceptually homogeneous. One strategy to model the first requirement is to extract features, generate a codebook dictionary of feature descriptors, and ask that distribution over the codebook (for foreground voxels) be consistent with the distribution induced by the expert-segmented foreground (on the same codebook). Putting this together with the homogeneity requirement serves to define the problem: segment a given DTI image (using MRFs, normalized cuts), while ensuring that the extracted foreground matches a known appearance model (over a bag of codebook features). The goal is related to recent work on simultaneous segmentation of two images called Cosegmentation [8, 9, 10, 11].

In the following sections, we formalize the problem and then present efficient segmentation methods. The **key contributions** of this paper are: **(i)** We propose a new algorithm for epitome-based graph-cuts segmentation, one which permits introduction of a bias to favor solutions that match a given epitome for regions of interest. **(ii)** We present an application to segmentation of specific structures in Diffusion Tensor Images of the human brain and provide experimental evidence that many structures of interest in Neuroscience can be extracted reliably from large 3-D DTI images. **(iii)** Our analysis provides a guarantee of a *constant* factor approximation ratio of 4. For a deterministic round-up strategy to obtain integral solutions, this approximation is provably tight.

## 2 Preliminaries

We provide a short overview of how image segmentation is expressed as finding the maximum likelihood solution to a Conditional or Markov Random Field function. Later, we extend the model to include an additional bias (or regularizer) so that the configurations that are consistent with an epitome of a structure of interest turn out to be *more* likely (than other possibly lower energy solutions).

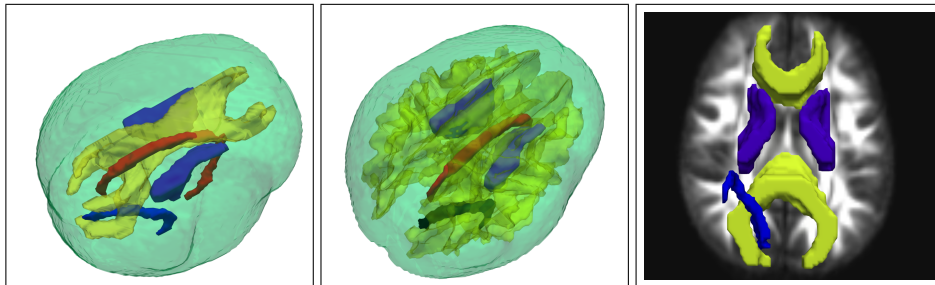

Figure 1: Specific white matter structures such as Corpus Callosum, Interior Capsules, and Cingulum Bundle are shown in 3D (left), within the entire white matter (center), and overlaid on a Fractional Anisotropy (FA) image slice (right). Our objective is to segment such structures from DTI images. Note that FA is a scalar anisotropy measure often used directly for WM/GM segmentation, since anisotropy is higher in white matter.

## 2.1 Markov Random Fields (MRF)

Markov Random Field based image segmentation approaches are quite popular in computer vision [12, 13] and neuroimaging [14]. A random field is assumed over the image lattice consisting of discrete random variables, $\mathbf{x} = \{x_1, \cdots, x_n\}$. Each $x_j \in \mathbf{x}, j \in \{1, \cdots, n\}$ takes a value from a finite label set, $\mathcal{L} = \{\mathcal{L}_1, \cdots, \mathcal{L}_m\}$. The set $\mathcal{N}_j = \{i | j \sim i\}$ lists the neighbors of $x_j$ on the adjacency lattice, denoted as $(j \sim i)$. A configuration of the MRF is an assignment of each $x_j$ to a label in $\mathcal{L}$. Labels represent distinct image segments; each configuration gives a segmentation, and the desired segmentation is the least energy MRF configuration. The energy is expressed as a sum of (1) individual data log-likelihood terms (cost of assigning $x_j$ to $\mathcal{L}_k \in \mathcal{L}$) and (2) pairwise smoothness prior (favor voxels with similar appearance to be assigned to the same label) [12, 15, 16]:

$$\min_{\mathbf{x}, \mathbf{z}} \quad \sum_{\mathcal{L}_k \in \mathcal{L}} \sum_{j=1}^{n} w_{jk} x_{jk} + \sum_{(i \sim j)} c_{ij} z_{ij} \tag{1}$$

$$\text{subject to} \quad |x_{ik} - x_{jk}| \le z_{ij} \quad \forall k \in \{1, \cdots, m\}, \quad \forall (i \sim j) \in \mathcal{N} \text{ where } i, j \in \{1, \cdots, n\}, \tag{2}$$

$$\mathbf{x} \text{ is binary of size } n \times m, \mathbf{z} \text{ is binary of size } |\mathcal{N}|, \tag{3}$$

where $w_{jk}$ is a unary term encoding the probability of $j$ being assigned to $\mathcal{L}_k \in \mathcal{L}$, and $c_{ij}$ is the pairwise smoothness prior (e.g., Generalized Potts model). The variable $z_{ij} = 1$ indicates that voxels $i$ and $j$ are assigned to *different* labels and $\mathbf{x}$ provides the assignment of voxel to labels (i.e., segments or regions). The problem is NP-hard but good approximation algorithms (including combinatorial methods) are known [16, 15, 17, 12]. Special cases (e.g., when $\mathbf{c}$ is convex) are known to be poly-time solvable [15]. Next, we discuss an interesting extension of MRF segmentation, namely Cosegmentation, which deals with the simultaneous segmentation of multiple images.

## 2.2 From Cosegmentation toward Epitome-based MRFs

Cosegmentation uses the observation that while *global* histograms of images of the same object (in different backgrounds) may differ, the histogram(s) of the respective foreground regions in the image pair (based on certain invariant features) remain relatively stable. Therefore, one may perform a concurrent segmentation of the images with a global constraint that enforces consistency between histograms of *only* the foreground voxels. We first construct a codebook of features $\mathcal{F}$ (e.g., using RGB intensities) for images $\mathcal{I}^{(1)}$ and $\mathcal{I}^{(2)}$; the histograms on this dictionary are:

$$\mathcal{H}^{(1)} = \{\mathcal{H}_1^{(1)}, \cdots, \mathcal{H}_\beta^{(1)}\} \text{ and } \mathcal{H}^{(2)} = \{\mathcal{H}_1^{(2)}, \cdots, \mathcal{H}_\beta^{(2)}\} \text{ ($b$ indexes the histogram bins)},$$

such that $\mathcal{H}_b^{(u)}(j) = 1$ if voxel $j \in \mathcal{I}^{(u)}$ is most similar to codeword $\mathcal{F}_b$, where $u \in \{1, 2\}$. If $\mathbf{x}^{(1)}$ and $\mathbf{x}^{(2)}$ denote the segmentation solutions, and $x_j^{(1)} = 1$ assigns voxel $j$ of $\mathcal{I}^{(1)}$ to the foreground, a measure of consistency between the foreground regions (after segmentation) is given by:

$$\sum_{b=1}^{\beta} \Psi \left( \langle \mathcal{H}_b^{(1)}, \mathbf{x}^{(1)} \rangle, \langle \mathcal{H}_b^{(2)}, \mathbf{x}^{(2)} \rangle \right). \tag{4}$$

where $\Psi(\cdot, \cdot)$ is a suitable similarity (or distance) function and $\langle \mathcal{H}_b^{(u)}, \mathbf{x}^{(u)} \rangle = \sum_{j=1}^{n} \mathcal{H}_b^{(u)}(j) x_j^{(u)}$, a count of the number of voxels in $\mathcal{I}^{(u)}$ (from $\mathcal{F}_b$) assigned to the foreground for $u \in \{1, 2\}$. Using (4) to regularize the segmentation objective (1) biases the model to favor solutions where the foregrounds match (w.r.t. the codebook $\mathcal{F}$), leading to more consistent segmentations.

The form of $\Psi(\cdot, \cdot)$ above has a significant impact on the hardness of the problem, and different ideas have been explored [8, 9, 10]. For example, the approach in [8] uses the $\ell_1$ norm to measure (and penalize) the variation, and requires a Trust Region based method for optimization. The sum of squared differences (SSD) function in [9] leads to partially optimal (half integral) solutions but requires solving a large linear program – infeasible for the image sizes we consider (which are orders of magnitude larger). Recently, [10] substituted $\Psi(\cdot, \cdot)$ with a so-called reward on histogram similarity. This *does* lead to a polynomial time solvable model, but requires the similarity function to be quite discriminative (otherwise offering a reward might be counter-productive in this setting).

## 3 Optimization Model

We start by using the sum of squared differences (SSD) as in [9] to bias the objective function and incorporate epitome awareness within the MRF energy in (1). However, unlike [9], where one

seeks a segmentation of *both* images, here we are provided the second histogram – the epitome (representation) of the specific region of interest. Clearly, this significantly simplifies the resultant Linear Program. Unfortunately, it remains computationally intractable for high resolution 3-D image volumes ($256^2 \times 128$) we consider here (the images are much larger than what is solvable by state of the art LP software, as in [9]). We propose a solution based on a combinatorial method, using ideas from some recent papers on Quadratic Pseudoboolean functions and their applications [18, 19]. This allows us to apply our technique on large scale image volumes, and obtain accurate results quite efficiently. Further, our analysis shows that we can obtain good constant-factor approximations (these are tight under mild conditions). We discuss our formulation next.

We first express the objective in (1) with an additional regularization term to penalize histogram dissimilarity using the sum of squared differences. This gives the following simple expression,

$$\min_{\mathbf{x},\mathbf{z}} \sum_{i \sim j} c_{ij} z_{ij}^{(1)} + \sum_{j=1}^{n} w_{j0}(1 - x_j^{(1)}) + \sum_{j=1}^{n} w_{j1} x_j^{(1)} + \lambda \sum_{b=1}^{\beta} (\langle \mathcal{H}_b^{(1)}, \mathbf{x}^{(1)} \rangle - \underbrace{\hat{\mathcal{H}}_b}_{\langle \mathcal{H}_b^{(2)}, \mathbf{x}^{(2)} \rangle})^2$$

Since the epitome (histogram) is provided, the second argument of $\Psi(\cdot, \cdot)$ in (4) is replaced with $\hat{\mathcal{H}}$, and $\mathbf{x}^{(1)}$ represents the solution vector for image $\mathcal{I}^{(1)}$. In addition, the term $w_{j0}$ (and $w_{j1}$) denote the unary cost of assigning voxel $j$ to the background (and foreground), and $\lambda$ is a user-specified tunable parameter to control the influence of the histogram variation. This yields

$$\min_{\mathbf{x},\mathbf{z}} \quad \sum_{i \sim j} c_{ij} z_{ij} + \sum_{j=1}^{n} w_{j0}(1 - x_j) + \sum_{j=1}^{n} w_{j1} x_j + \lambda \sum_{b=1}^{\beta} \left( \langle \mathcal{H}_b, \mathbf{x} \rangle^2 - 2\langle \mathcal{H}_b, \mathbf{x} \rangle \hat{\mathcal{H}}_b + \underbrace{\hat{\mathcal{H}}_b^2}_{\text{constant}} \right)$$

subject to $\quad |x_i - x_j| \leq z_{ij} \quad \forall (i \sim j)$ where $i, j \in \{1, \cdots, n\}$, and $\quad \mathbf{x}, \mathbf{z}$ is binary, $\qquad$ (5)

The last term in (5) is constant. So, the model reduces to

$$\min_{\mathbf{x},\mathbf{z}} \quad \sum_{i \sim j} c_{ij} z_{ij} + \sum_{j=1}^{n} w_{j0}(1 - x_j) + \sum_{j=1}^{n} w_{j1} x_j + \lambda \sum_{b=1}^{\beta} \left( \sum_{j=1}^{n} \sum_{l=1}^{n} \mathcal{H}_b(j) \mathcal{H}_b(l) x_j x_l - 2 \sum_{j=1}^{n} \mathcal{H}_b(j) x_j \hat{\mathcal{H}}_b \right)$$

s.t. $\quad |x_i - x_j| \leq z_{ij} \quad \forall (i \sim j)$ where $i, j \in \{1, \cdots, n\}$, and $\quad \mathbf{x}, \mathbf{z}$ is binary, $\qquad$ (6)

Observe that (6) can be expressed as a special case of the general form, $\Gamma(x_1, \cdots, x_n) = \sum_{S \subset U} \phi_S \prod_{j \in S} x_j$ where $U = \{1, \cdots, n\}$, $\mathbf{x} = (x_1, \cdots, x_n) \in \mathbb{B}^n$ is a binary vector, $S$ is a subset of $U$, and $\phi_S$ denotes the coefficient of $S$. Such a function $\Gamma : \mathbb{B}^n \mapsto \mathbb{R}$ is called a pseudo-Boolean function [18]. If the cardinality of $S$ is no more than two, the corresponding form is

$$\Gamma(x_1, x_2, \cdots, x_n) = \sum_{j} \phi_j x_j + \sum_{(i,j)} \phi_{ij} x_i x_j$$

These functions are called Quadratic Pseudo-Boolean functions (QPB). In general if the objective permits a representation as a QPB, an upper (or lower) bound can be derived using roof (or floor) duality [18], recently utilized in several papers [19, 20, 21]. Notice that the function in (6) is a QPB because it has at most two variables in each term in the expansion. An advantage of the model derived above is that (pending some additional adjustments) we will be able to leverage an extensive existing combinatorial machinery to solve the problem. We discuss these issues in more detail next.

## 4  Reparameterization and Graph Construction

Now we discuss a graph construction to optimize the above energy function by computing a maximum flow/minimum cut. We represent each variable as a pair of literals, $x_j$ and $\bar{x}_j$, which corresponds to a pair of nodes in a graph $\mathcal{G}$. Edges are added to $\mathcal{G}$ based on various terms in the corresponding QPB. The min-cut computed on $\mathcal{G}$ will determine the assignments of variables to 1 (or 0), i.e., foreground/background assignment. Depending on how the nodes for a pair of literals are partitioned, we either get "persistent" integral solutions (same as in optimal) and/or obtain variables assigned $\frac{1}{2}$ (half integral) values and need additional rounding to obtain a $\{0, 1\}$ solution.

We will first reparameterize the coefficients in our objective as a vector denoted by $\mathbf{\Phi}$. More specifically, we express the energy by collecting the unary and pairwise costs in (6) as the coefficients of the linear and quadratic variables. For a voxel $j$, we denote the unary coefficient as $\mathbf{\Phi}_j$ and for a pair of voxels $(i, j)$ we give their corresponding coefficients as $\mathbf{\Phi}_{ij}$. For presentation, we show

| Voxel pairs $(i,j)$ | $i \sim j, i \not\cong j$ | $i \not\sim j, i \cong j$ | $i \sim j, i \cong j$ |
|---|---|---|---|
| $(v_i \rightarrow v_j), (\bar{v}_j \rightarrow \bar{v}_i)$ | $\frac{1}{2}c_{ij}$ | $0$ | $\frac{1}{2}c_{ij}$ |
| $(v_j \rightarrow v_i), (\bar{v}_i \rightarrow \bar{v}_j)$ | $\frac{1}{2}c_{ij}$ | $0$ | $\frac{1}{2}c_{ij}$ |
| $(\bar{v}_j \rightarrow v_i), (\bar{v}_i \rightarrow v_j)$ | $0$ | $\frac{1}{2}\lambda$ | $\frac{1}{2}\lambda$ |

Table 1: Illustration of edge weights introduced in the graph for voxel pairs.

spatial adjacency as $i \sim j$, and if $i$ and $j$ share a bin in the histogram we denote it as $i \cong j$, i.e., $\exists b : \mathcal{H}_b(i) = \mathcal{H}_b(j) = 1$. The definition of the *pairwise* costs will include the following scenarios:

$$\mathbf{\Phi}_{ij} = \begin{cases} c_{ij} & \text{if } i \sim j \quad \text{and} \quad i \not\cong j \quad \text{and} \quad (i,j) \text{ assigned to different labels} \\ \lambda & \text{if } i \not\sim j \quad \text{and} \quad i \cong j \quad \text{and} \quad (i,j) \text{ assigned to foreground} \\ c_{ij} & \text{if } i \sim j \quad \text{and} \quad i \cong j \quad \text{and} \quad (i,j) \text{ assigned to different labels} \\ \lambda & \text{if } i \sim j \quad \text{and} \quad i \cong j \quad \text{and} \quad (i,j) \text{ assigned to foreground} \end{cases} \tag{7}$$

The above cases enumerate three possible relationships between a pair of voxels $(i,j)$: **(i)** $(i,j)$ are spatial neighbors but *not* bin neighbors; **(ii)** $(i,j)$ are bin neighbors, but *not* spatial neighbors; **(iii)** $(i,j)$ are bin neighbors *and* spatial neighbors. In addition, the cost is also a function of label assignments to $(i,j)$. Note that we assume $i \neq j$ above since if $i = j$, we can absorb those costs in the unary terms (because $x_i \cdot x_i = x_i$). We define the *unary* costs for each voxel $j$ next.

$$\mathbf{\Phi}_j = \begin{cases} w_{j0} & \text{if } j \text{ is assigned to background} \\ w_{j1} + \lambda - 2\lambda\hat{\mathcal{H}}_b & \text{if } j \text{ is assigned to foreground and } \exists b : \mathcal{H}_b(i) = 1 \end{cases} \tag{8}$$

With the reparameterization given as $\mathbf{\Phi} = [\mathbf{\Phi}_j \ \mathbf{\Phi}_{ij}]^T$ done, we follow the recipe in [18, 22] to construct a graph (briefly summarized below). For each voxel $j \in \mathcal{I}$, we introduce two nodes, $v_j$ and $\bar{v}_j$. Hence, the size of the graph is $2|\mathcal{I}|$. We also have two special nodes $s$ and $t$ which denote the source and sink respectively. We connect each node to the source and/or the sink based on the unary costs, assuming that the source (and sink) partitions correspond to foreground (and background). The source is connected to the node $v_j$ with weight, $\frac{1}{2}(w_{j1} + \lambda - 2\lambda\hat{\mathcal{H}}_b)$, and to node $\bar{v}_j$ with weight $\frac{1}{2}w_{j0}$. Nodes $v_j$ and $\bar{v}_j$ are in turn connected to the sink with costs $\frac{1}{2}w_{j0}$ and $\frac{1}{2}(w_{j1} + \lambda - 2\lambda\hat{\mathcal{H}}_b)$ respectively. These edges, if saturated in a max-flow, count towards the node's unary cost. Edges between node pairs (except source and sink) give pairwise terms of the energy. These edge weights (see Table 1) quantify all possible relationships of pairwise voxels and label assign-

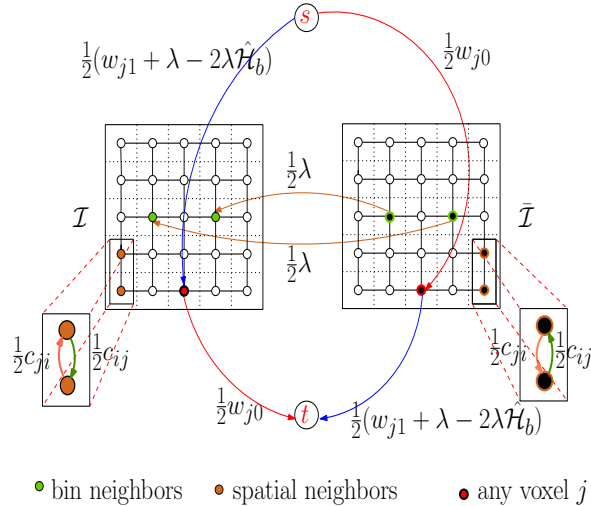

Figure 2: A graph to optimize (6). Nodes in the left box represents $v_j$; nodes in the right box represent $\bar{v}_j$. Colors indicate spatial neighbors (orange) or bin neighbors (green).

ments (Fig. 2). A maximum flow/minimum cut procedure on this graph gives a solution to our problem. After the cut, each node (for a voxel) is connected either to the source set or to the sink set. Using this membership, we can obtain a final solution (i.e., labeling) as follows.

$$x_j = \begin{cases} 0 & \text{if } v_j \in s, \bar{v}_j \in t \\ 1 & \text{if } v_j \in t, \bar{v}_j \in s \\ \frac{1}{2} & \text{otherwise} \end{cases} \tag{9}$$

A property of the solution obtained by (9) is that the variables assigned $\{0, 1\}$ values are "persistent", i.e., they are the *same* in the optimal integral solution to (6). This means that the solution from the algorithm above is partially optimal [18, 20]. We now only need to find an assignment for the $\frac{1}{2}$ variables (to 0 or 1) by rounding. The rounding strategy and analysis is presented next.

## 5 Rounding and Approximation analysis

In general, any reasonable heuristic can be used to round $\frac{1}{2}$-valued variables to 0 or 1 (e.g., we can solve for and obtain a segmentation for *only* the $\frac{1}{2}$-valued variables *without* the additional bias). Our

experiments later make use of such a heuristic. The approximation analysis below, however, is based on a more conservative scheme of rounding *all* $\frac{1}{2}$-valued variables up to $1$. We only summarize our main results here, the longer version of the paper includes details.

A 2-approximation for the objective function (without the epitome bias) is known [16, 12]. The rounding above gives a constant factor approximation.

**Theorem 1** *The rounding strategy described above gives a feasible solution to Problem (6). This solution is a factor $4$ approximation to (6). Further, the approximation ratio is tight for this rounding.*

## 6 Experimental Results

**Overview.** We now empirically evaluate our algorithm for extracting specific structures of interest from DTI data, focusing on **(1)** Corpus Callosum (CC), and **(2)** Interior Capsule (IC) as representative examples. Our experiments were designed to answer the following main questions: **(i)** Can the model reliably and accurately identify the structures of interest? Note that general-purpose white matter segmentation methods do not extract *specific* regions (which is often obtained via intensive interactive methods instead). Solutions from our algorithm, if satisfactory, can be used directly for analysis or as a warm-start for user-guided segmentations for additional refinement. **(ii)** Does segmentation with a bias for fidelity with epitomes offer advantages over training a classifier on the *same* features? Clearly, the latter scheme will work nicely if the similarity between foreground/background voxels is sufficiently discriminative. Our experiments provide evidence that epitomes indeed offer advantages. **(iii)** Finally, we evaluate the advantages of our method in terms of relative effort expended by a user performing interactive extraction of CC and IC from 3-D volumes.

**Data and Setup.** We acquired $25$ Diffusion Tensor brain images in $12$ non-collinear diffusion encoding directions (and one $b = 0$ reference image) with diffusion weighting factor of $b = 1000s/\mathrm{mm}^2$. Standard image processing included correcting for eddy current related distortion, distortion from field inhomogeneities (using field maps), and head motion. From this data, the tensor elements were estimated using standard toolboxes (Camino [23]). The images were then hand-segmented (slice by slice) by experts to serve as the gold standard segmentation. Within a leave one out cross validation scheme, we split our set into training ($24$ images) and test set (hold out image). Epitomes were constructed using training data (by averaging tensor volumes and generating feature codeword dictionaries), and then specific structures in the hold out image were segmented using our model. Codewords used for the epitome also served to train a SVM classifier (on training data), which was then used to label voxels as foreground (part of structure of interest) or background, in the hold-out image. We present the mean of segmentation accuracy over $25$ realizations.

**WM/GM DTI segmentation.** To briefly elaborate on **(i)** above, we note that most existing DTI segmentation algorithms in the literature [24] focus on segmenting the *entire* white-matter (WM) from gray-matter (GM) where as the focus here is to extract specific structure *within* the WM pathways, to facilitate the type of analysis being pursued in neuroscience studies

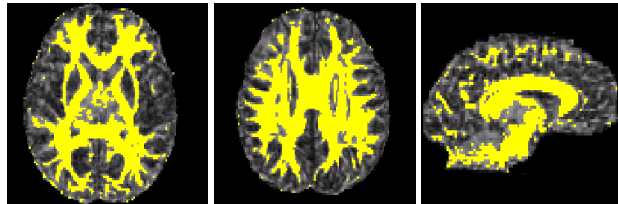

Figure 3: WM/GM segmentation (without epitomes) from standard toolkits, overlaid on FA maps (axial, sagittal views shown).

[25, 2]. Fig. 3 shows results of a DTI image WM segmentation. Such methods segment WM well but are not designed to identify different components within the WM. Certain recent works [26] have reported success in identifying structures such as the cingulum bundle if a good population specific atlas is available (here, one initializes the segmentation by a sophisticated registration procedure).

**Dictionary Generation.** A suitable codebook of features (i.e., $\mathcal{F}$ from §2.2) is essential to modulate the segmentation (with an uninformative histogram, the process degenerates to a ordinary segmentation without epitomes). Results from our preliminary experiments suggested that the codeword generation must be informed by the properties/characteristics of Diffusion Tensor images. While general purpose feature extractors or interest-point detectors from Vision cannot be directly applied to tensor data, our simple scheme below is derived from these ideas. Briefly, by first setting up a neighborhood region around each voxel, we evaluate the local orientation context and shape in-

formation from the principal eigen vectors and eigen values of tensors at each neighboring voxel. Similar to Histogram of Oriented Gradients or SIFT, each neighboring voxel casts a vote for the primary eigen vector orientation (weighted by its eigen value), which encodes the distribution of tensor orientations in a local neighborhood around the voxel, as a feature vector. These feature vectors are then clustered, and each voxel is 'assigned' to its closest codeword/feature to give $\mathcal{H}^{(u)}$. Certain adjustments are needed for structurally sparse regions close to periphery of the brain surface, where we use all primary eigen vectors in a (larger) neighborhood window. This dictionary generation is not rotationally invariant since the orientation of the eigen-vectors are used. Our literature review suggests that there is no 'accepted' strategy for feature extraction from tensor-valued images. While the problem is interesting, the procedure here yields reasonable results for our purpose. We acknowledge that improvements may be possible using more sophisticated approaches.

**Implementation Details.** Our implementation in C++ was interfaced with a QPB solver [22, 18]. We used a distance measure proposed in DTI-TK [23] which is popular in the neuroimaging literature, to obtain a similarity measure between tensors. The unary terms for the MRF component were calculated as the least DTI-TK metric distance between the voxel and a set of labels (generated by sampling from foreground in the training data). Pairwise smoothness terms were calculated using a spatial neighborhood of 18 neighbors. The parameter $\lambda$ was set to 10 for all runs.

### 6.1 Results: User guided interactive segmentation, Segmentation with Epitomes and SVMs

**User study for interactive segmentation.** To assess the amount of effort expended in obtaining a good segmentation of the regions of interest in an interactive manner, we set up a user study with two users who were familiar with (but not experts in) neuroanatomy. The users were presented with the ground truth solution for each image. The user provided "scribbles" denoting foreground/background regions, which were incorporated into the segmentation via must-link/cannot-link constraints. Ignoring the time required for segmentation, typically 20-40 seeds were needed for each 2-D slice/image to obtain results close to ground-truth segmentations, which required $\sim 60s$ of user participation per 3-4 slices. Representative results are presented in Figs. 4–5 (column 5).

**Results from SVM and our model.** For comparison, we trained a SVM classifier on the same set of voxel-codewords used for the epitomes. For training, feature vectors for foreground/background voxels from the training images were used, and the learnt function was used to classify voxels in the hold-out image. Representative results are presented in Figs. 4–5, overlaid on 2-D slices of Fractional Anisotropy. We see good consistency between our solutions and the ground truth in Figs. 4–5 where as the SVM results seem to oversegment, undersegment or pick up erroneous regions with similar contextual appearance to some voxels in the epitome. It is true that such a classification experiment with better (more discriminative) features will likely perform better; however, it is not clear how to reliably extract good quality features from tensor valued images. The results also suggest that our model exploits the epitome of such features rather well within a segmentation criterion.

**Quantitative Summary.** For quantitative evaluations, we computed the Dice Similarity coefficient between the segmentation solutions $\mathcal{A}$ and the expert segmentation $\mathcal{B}$, given as $\frac{2(\mathcal{A} \cap \mathcal{B})}{|\mathcal{A}|+|\mathcal{B}|}$. On CC and IC, the similarity coefficient of our solutions were $0.62 \pm 0.04$ and $0.57 \pm 0.05$ respectively. The corresponding values for the SVM segmentation were $0.28 \pm 0.06$ and $0.15 \pm 0.02$ respectively. Hence, the null hypothesis using a two sample $t$-test can be rejected at $\alpha = 0.01$ (significance level). The running time of our algorithm was comparable to the running times of SVM using Shogun (a subset of voxels were used for training). It took $\sim 2$ mins for our algorithm to solve the network flow on the graph, and $< 4$ mins to read in the images and construct the graph. While the segmentation results from the user-guided interactive segmentation are marginally better than ours, the user study above indicates that a significant level of interaction is required, which is already difficult for large 3-D volumes and becomes impractical for neuroimaging studies with tens of image volumes.

## 7 Discussion and Conclusions

We present a new combinatorial algorithm for segmenting specific structures from DTI images. Our goal is to *segment* the structure while maintaining consistency with an epitome of the structure, generated from expert segmented images (note that this is different from top-down segmentation approaches [27], and algorithms which use a parametric prior [28, 11]). We see that direct application of max-margin methods does not yield satisfactory results, and inclusion of a segmentation-specific objective function seems essential. Our derived model can be optimized using a network flow pro-

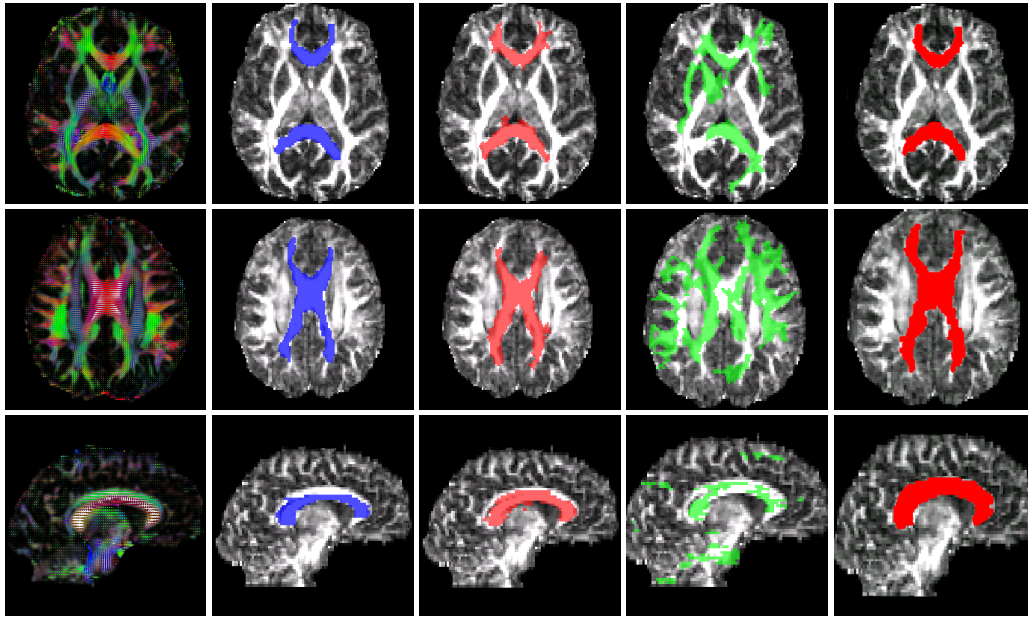

Figure 4: A segmentation of the Corpus Callosum overlaid on FA maps. Rows refer to axial and sagittal views. Columns: (1) Tensors. (2) Ground truth. (3) Our solutions. (4) SVM results. (5) User-guided segmentation.

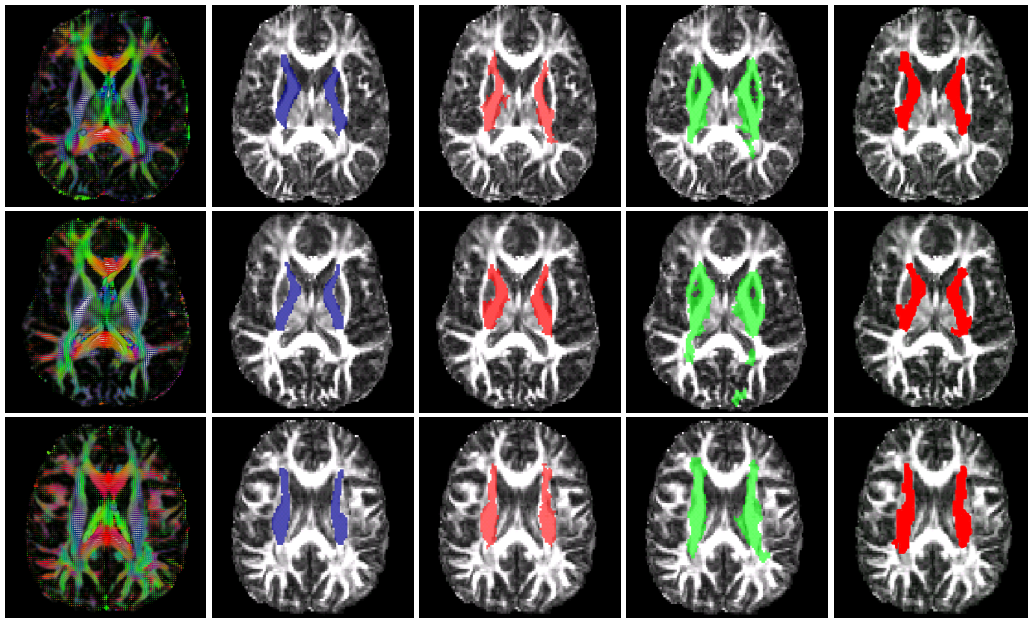

Figure 5: A segmentation of the Interior Capsules overlaid on FA maps. Rows correspond to axial views. Columns: (1) Tensors. (2) Ground truth. (3) Our Solutions. (4) SVM results. (5) User-guided segmentation.

cedure. We also prove a $4$ factor approximation ratio, which is tight for the proposed rounding mechanism. We present experimental evaluations on a number of large scale image volumes which shows that the approach works well, and is also computationally efficient (2-3 mins). Empirical improvements seem possible by designing better methods of feature extraction from tensor-valued images. The model may serve to incorporate epitomes for general segmentation problems on other images as well. In summary, our approach shows that many structures of interest in neuroimaging can be accurately extracted from DTI data.

## Footnotes

*Supported by AG034315 (Singh), MH62015 (Alexander), UW ICTR (1UL1RR025011), and UW ADRC (AG033514). Hinrichs and Adluru are supported by UW-CIBM funding (via NLM 5T15LM007359). Thanks to Richie Davidson for assistance with the data, and Anne Bartosic and Chad Ennis for ground truth indications. The authors thank Lopamudra Mukherjee, Moo K. Chung, and Chuck Dyer for discussions and suggestions.

# References

[1] J. Burns, D. Job, M. E. Bastin, et al. Structural disconnectivity in schizophrenia: a diffusion tensor magnetic resonance imaging study. *The British J. of Psychiatry*, 182(5):439–443, 2003. 1

[2] A. Pfefferbaum and E. Sullivan. Microstructural but not macrostructural disruption of white matter in women with chronic alcoholism. *Neuroimage*, 15(3):708–718, 2002. 1, 6

[3] T. Liu, H. Li, K. Wong, et al. Brain tissue segmentation based on DTI data. *Neuroimage*, 38:114–123, 2007. 2

[4] Z. Wang and B. Vemuri. DTI segmentation using an information theoretic tensor dissimilarity measure. *Trans. on Med. Imaging*, 24:1267–1277, 2005. 2

[5] P. A. Yushkevich, H. Zhang, T. J. Simon, and J. C. Gee. Structure-specific statistical mapping of white matter tracts using the continuous medial representation. In *Proc. of MMBIA*, 2007. 2

[6] N. Lawes, T. Barrick, V. Murugam, et al. Atlas based segmentation of white matter tracts of the human brain using diffusion tensor tractography and comparison with classical dissection. *Neuroimage*, 39:62–79, 2008. 2

[7] C. B. Goodlett, T. P. Fletcher, J. H. Gilmore, and G. Gerig. Group analysis of DTI fiber tract statistics with application to neurodevelopment. *Neuroimage*, 45(1):S133 – S142, 2009. 2

[8] C. Rother, T. Minka, A. Blake, and V. Kolmogorov. Cosegmentation of image pairs by histogram matching: Incorporating a global constraint into MRFs. In *Comp. Vision and Pattern Recog.*, 2006. 2, 3

[9] L. Mukherjee, V. Singh, and C. Dyer. Half-integrality based algorithms for cosegmentation of images. In *Comp. Vision and Pattern Recog.*, 2009. 2, 3, 4

[10] D. Hochbaum and V. Singh. An efficient algorithm for co-segmentation. In *Intl. Conf. on Comp. Vis.*, 2009. 2, 3

[11] D. Batra, A. Kowdle, D. Parikh, et al. icoseg: Interactive co-segmentation with intelligent scribble guidance. In *Comp. Vision and Patter Recog.*, 2010. 2, 7

[12] Y. Boykov, O. Veksler, and R. Zabih. Fast approximate energy minimization via graph cuts. *Trans. on Pattern Anal. and Machine Intel.*, 23(11):1222–1239, 2001. 3, 6

[13] V. Kolmogorov, Y. Boykov, and C. Rother. Applications of parametric maxflow in Computer Vision. In *Intl. Conf. on Comp. Vision*, 2007. 3

[14] Y. T . Weldeselassie and G. Hamarneh. DT -MRI segmentation using graph cuts. In *Medical Imaging: Image Processing*, volume 6512 of *Proc. SPIE*, 2007. 3

[15] D. Hochbaum. An efficient algorithm for image segmentation, markov random fields and related problems. *J. of the ACM*, 48(4):686–701, 2001. 3

[16] J. Kleinberg and E. Tardos. Approximation algorithms for classification problems with pairwise relationships: Metric partitioning and markov random fields. *J. of the ACM*, 49(5):616–639, 2002. 3, 6

[17] H. Ishikawa. Exact optimization for markov random fields with convex priors. *Trans. on Pattern Anal. and Machine Intel*, 25(10):1333–1336, 2003. 3

[18] E. Boros and P. Hammer. Pseudo-Boolean optimization. *Disc. Appl. Math.*, 123:155–225, 2002. 4, 5, 7

[19] C. Rother, V. Kolmogorov, V. Lempitsky, and M. Szummer. Optimizing binary mrfs via extended roof duality. In *Comp. Vision and Pattern Recog.*, 2007. 4

[20] P. Kohli, A. Shekhovtsov, C. Rother, V. Kolmogorov, et al. On partial optimality in multi-label MRFs. In *Intl. Conf. on Machine learning*, 2008. 4, 5

[21] A. Raj, G. Singh, and R. Zabih. MRFs for MRIs: Bayesian reconstruction of MR images via graph cuts. In *Comp. Vision and Pattern Recog.*, 2006. 4

[22] V. Kolmogorov and C. Rother. Minimizing nonsubmodular functions with graph cuts-a review. *Trans. on Pattern Anal. and Machine Intel.*, 29(7):1274, 2007. 5, 7

[23] H. Zhang, P. A. Yushkevich, D. C. Alexander, and J. C. Gee. Deformable registration of diffusion tensor MR images with explicit orientation optimization. *Medical Image Analysis*, 10:764–785, 2006. 6, 7

[24] M. Rousson, C. Lenglet, and R. Deriche. Level set and region based surface propagation for diffusion tensor MRI segmentation. In *Proc. of CVAMIA-MMBIA*, volume 3117 of *LNCS*, pages 123–134, 2004. 6

[25] S. M. Smith, M. Jenkinson, H. Johansen-Berg, et al. Tract-based spatial statistics: Voxelwise analysis of multi-subject diffusion data. 31:1487–1505, 2006. 6

[26] S. Awate, H. Zhang, and Gee. J. A fuzzy, nonparametric segmentation framework for DTI and MRI analysis with applications to DTI tract extraction. *Trans. on Med. Imaging*, 26(11):1525–1536, 2007. 6

[27] E. Borenstein, E. Sharon, and S. Ullman. Combining top-down and bottom-up segmentation. In *Comp. Vision and Pattern Recognition Workshop*, 2004. 7

[28] C. Jingyu, Y. Qiong, W. Fang, et al. Transductive object cutout. In *Comp. Vision and Pattern Recog.*, 2008. 7

